# Hierarchical Matching Pursuit for Image Classification: Architecture and Fast Algorithms

**Liefeng Bo**
University of Washington
Seattle WA 98195, USA

**Xiaofeng Ren**
ISTC-Pervasive Computing Intel Labs
Seattle WA 98195, USA

**Dieter Fox**
University of Washington
Seattle WA 98195, USA

## Abstract

Extracting good representations from images is essential for many computer vision tasks. In this paper, we propose hierarchical matching pursuit (HMP), which builds a feature hierarchy layer-by-layer using an efficient matching pursuit encoder. It includes three modules: batch (tree) orthogonal matching pursuit, spatial pyramid max pooling, and contrast normalization. We investigate the architecture of HMP, and show that all three components are critical for good performance. To speed up the orthogonal matching pursuit, we propose a batch tree orthogonal matching pursuit that is particularly suitable to encode a large number of observations that share the same large dictionary. HMP is scalable and can efficiently handle full-size images. In addition, HMP enables linear support vector machines (SVM) to match the performance of nonlinear SVM while being scalable to large datasets. We compare HMP with many state-of-the-art algorithms including convolutional deep belief networks, SIFT based single layer sparse coding, and kernel based feature learning. HMP consistently yields superior accuracy on three types of image classification problems: object recognition (Caltech-101), scene recognition (MIT-Scene), and static event recognition (UIUC-Sports).

## 1   Introduction

Visual recognition is a major focus of research in computer vision, machine learning, and robotics. Many real world vision systems fundamentally rely on the ability to recognize object instances, categories, scenes, and activities. In the past few years, more and more people have realized that the core of building recognition systems is to learn meaningful representations (features) from high-dimensional observations such as images and videos. A growing amount of research on visual recognition has focused on learning rich features using modern machine learning methods.

Deep belief nets [9] built a hierarchy of features by greedily training each layer separately using the restricted Boltzmann machine. The learned weights are then used to initialize multi-layer feed-forward networks that further adjust the weights to the task at hand using supervision. To handle full-size images, Lee et al. [16] proposed convolutional deep belief networks (CDBN) that use a small receptive field and share the weights between the hidden and visible layers among all locations in an image. Invariant predictive sparse decomposition [11, 13] used feed-forward neural networks to approximate sparse codes generated by sparse coding and avoided solving computationally expensive optimizations at runtime. Deconvolutional networks [26] reconstructed images using a group of latent feature maps in a convolutional way under a sparsity constraint. A fast optimization algorithm was introduced to solve the resulting sparse coding problem. These approaches

have been shown to yield competitive performance with the SIFT based bag-of-visual-words model on object recognition benchmarks such as Caltech-101.

Recent research has shown that single layer sparse coding on top of SIFT features works surprisingly well [15, 24, 23, 5, 6]. Yang et al. [24] proposed a single layer feature learning model ScSPM that uses SIFT features as the input to sparse coding instead of raw image patches. Their experiments have shown that this approach outperforms the classical bag-of-visual-words model and convolutional deep belief networks, and achieves the state-of-the-art performance on many image classification benchmarks. Wang et al. presented a fast implementation of local coordinate coding [23] that obtains sparse representations of SIFT features by performing local linear embedding on several nearest visual words in the codebook. Boureau et al. [5] compared many feature learning algorithms, and found that the SIFT based sparse coding in conjunction with max pooling performs remarkably well, and the macrofeatures can boost recognition performance further. Coates and Ng [6] evaluated many single layer feature learning systems by decomposing feature learning algorithms into training and encoding phases, and suggested that the choice of architecture and encoder is the key to a successful feature learning system. Very recently, Yu et al. [25] showed that hierarchical sparse coding (HSC) at pixel level achieves similar performance with SIFT based sparse coding.

However, single layer sparse coding heavily depends on hand-crafted SIFT features. It is desirable to develop efficient and effective algorithms to learn features from scratch. Motivated by the recent work on deep networks, in this work we propose hierarchical matching pursuit (HMP) that uses the matching pursuit encoder to build a feature hierarchy layer by layer. The matching pursuit encoder consists of three modules: batch tree orthogonal matching pursuit coding, spatial pyramid max pooling, and contrast normalization. We discuss the architecture of HMP, and show that spatial pyramid max pooling, contrast normalization, and hierarchical structure are key components to learn good representations for recognition. We further present batch tree orthogonal matching pursuit that is able to speed up the search of sparse codes significantly when a large number of observations share the same dictionary. Our CPU implementation of HMP can extract the features from a typical $300 \times 300$ image in less than one second. Our experiments on object recognition, scene recognition, and static event recognition confirm that HMP yields better accuracy than hierarchical feature learning, SIFT based single layer sparse coding, and many other state-of-the-art image classification algorithms on standard datasets. To the best of our knowledge, this is the first work to show that learning features from the pixel level significantly outperforms those approaches built on top of hand-crafted SIFT.

## 2  Hierarchical Matching Pursuit

In this section, we introduce hierarchical matching pursuit. We first show how K-SVD is used to learn the dictionary. We then propose the matching pursuit encoder, and investigate its architecture and fast algorithms to compute sparse codes. Finally, we discuss how to build hierarchical matching pursuit based on the matching pursuit encoder.

### 2.1  Dictionary Learning with K-SVD

K-SVD is a simple and efficient dictionary learning algorithm developed by Aharon et al. [1, 21]. K-SVD generalizes the idea of K-Means and updates the dictionary sequentially. Given a set of $h$-dimensional observations $Y = [y_1, \cdots, y_n] \in R^{h \times n}$ (image patches in our case), K-SVD learns a dictionary $D = [d_1, \cdots, d_m] \in R^{h \times m}$, where $d_i$ is called a filter (or atom), and an associated sparse code matrix $X = [x_1, \cdots, x_n] \in R^{m \times n}$ by minimizing the following reconstruction error

$$\min_{D, X} \|Y - DX\|_F^2 \qquad s.t. \ \forall i, \ \|x_i\|_0 \le K \qquad (1)$$

where the notation $\|A\|_F$ denotes the Frobenius norm, $x_i$ are the columns of $X$, the zero-norm $\| \cdot \|_0$ counts the non-zero entries in the sparse code $x_i$, and $K$ is the sparsity level, which bounds the number of the non-zero entries.

This optimization problem can be solved in an alternating manner. In the first stage, $D$ is fixed, and only the sparse code matrix is optimized. This problem can be decoupled to $n$ simpler sub-problems

$$\min_{x_i} \|y_i - Dx_i\|^2 \qquad s.t. \ \|x_i\|_0 \le K \qquad (2)$$

---

**Algorithm 1**: Batch Orthogonal Matching Pursuit (BOMP)

    1. Input: Dictionary $D$, observation $y$, and the desired sparsity level $K$

    2. Output: Sparse code $x$ such that $y \approx Dx$

    3. Initialization: $I = \emptyset$, $\alpha^0 = D^\top y$, $G = D^\top D$, and $x = 0$

    4. For $k = 1 : K$

        5.    Selecting the new filter: $\overline{k} = \mathrm{argmax}_k |\alpha_k|$

        6.    $I = I \cup \overline{k}$

        7.    Updating the sparse code: $x_I = G_{II}^{-1}\alpha_I^0$

        8.    Updating $\alpha$: $\alpha = \alpha^0 - G_I x_I$

    9. End

---

This optimization problem is combinational and highly non-convex, but its approximate solution can be found by the orthogonal matching pursuit discussed in the next section. In the second stage, the dictionary $D$ and its associated sparse coefficients are updated simultaneously by the Singular Value Decomposition (SVD). For a given filter $k$, the quadratic term in (1) can be rewritten as

$$\|Y - DX\|_F^2 = \|Y - \sum_{j \neq k} d_j x_j^\top - d_k x_k^\top\|_F^2 = \|E_k - d_k x_k^\top\|_F^2 \tag{3}$$

where $x_j^\top$ are the rows of $X$, and $E_k$ is the residual matrix for the $k$-th filter. The optimal $d_k$ and $x_k^\top$ can be obtained by performing SVD of the matrix $E_k$. To avoid the introduction of new non-zero entries in the sparse code matrix $X$, the update process only uses the observations whose sparse codes have used the $k$-th filter (the $k$-th entry of the associated sparse code is non-zero). When the sparsity level $K$ is set to be 1 and the sparse code matrix is forced to be a binary(0/1) matrix, K-SVD exactly reproduces the K-Means algorithm.

## 2.2 Matching Pursuit Encoder

Our matching pursuit encoder consists of three modules: batch tree orthogonal matching pursuit, spatial pyramid max pooling, and contrast normalization.

**Batch Tree Orthogonal Matching Pursuit.** The orthogonal matching pursuit (OMP) [19] computes an approximate solution for the optimization problem Eq.(2) in a greedy style. At each step, it selects the filter with the highest correlation to the current residual. At the first step, the residual is exactly the observation. Once the new filter is selected, the observation is orthogonally projected to the span of all the previously selected filters and the residual is recomputed. This procedure is repeated until the desired $K$ filters are selected. The quantities in the sparse code update need not be computed from scratch. The vector $D_I^\top y$ can be incrementally updated by simply appending a new entry $D_{\overline{k}}^\top y$, where $D_I$ denotes the sub-matrix of $D$ containing the columns indexed by $I$. The inversion of the matrix $(D_I^\top D_I)^{-1}$ can be obtained using a progressive Cholesky factorization that updates the matrix inversion incrementally.

In our application, sparse codes for a large number of image patches are computed by the same dictionary. The total cost of orthogonal matching pursuit can be reduced by batch orthogonal matching pursuit (BOMP) (Algorithm 1) that pre-computes some quantities [7, 22]. The key finding is that filter selection, the most expensive step, doesn't require $x$ and $r$ explicitly. Let $\alpha = D^\top r$, we have

$$\alpha = D^\top r = D^\top(y - D_I(D_I^\top D_I)^{-1}D_I^\top y) = \alpha^0 - G_I G_{II}^{-1}\alpha_I^0 \tag{4}$$

where we have set $\alpha^0 = D^\top y$ and $G = D^\top D$, and $G_{II}$ is the sub-matrix of $G$ containing the rows indexed by $I$ and the columns indexed by $I$. Equation (4) indicates that if $\alpha^0$ and $G$ are pre-computed, the cost of updating $\alpha$ is $O(mK)$, instead of $O(mh)$. In orthogonal matching pursuit, we have $K \leq h$ since the $h$ filters allow us to exactly reconstruct the observations. Note that the cost of searching sparse codes quickly dominates that of the pre-computation as observations increase. When using an overcomplete dictionary, $K$ is usually much less than $h$. In our experiments, $K$ is 10 and $h$ is several hundreds in the second layer of HMP, and we have observed significant speedup (Section 3) over orthogonal matching pursuit.

---
**Algorithm 2**: Batch Tree Orthogonal Matching Pursuit (BTOMP)
1. Input: Dictionary $D$, Centers $C$, observation $y$, and the desired sparsity level $K$
2. Output: Sparse code $x$ such that $y \approx Dx$
3. Initialization: $I = \emptyset$, $r = y$, $\alpha = \alpha^0 = C^\top y$, $B = C^\top D$, and $x = 0$
4. For $k = 1 : K$
5.     Choosing the sub-dictionary $g_j$: $j = \operatorname{argmax}_k |\alpha_k|$
6.     Selecting the new filter: $\overline{k} = \operatorname{argmax}_{k \in g_j} |d_k^\top r|$
7.     $I = I \cup \overline{k}$
8.     Updating the sparse code: $x_I = (D_I^\top D_I)^{-1} D_I^\top y$
9.     Updating $\alpha$: $\alpha = \alpha^0 - B_I x_I$
10.    Computing the residual: $r = y - D_I x_I$
11. End
---

Pre-computing $G$ takes $O(m^2 h)$ time and $O(m^2)$ memory, which becomes infeasible for a very large dictionary. To overcome this problem, we propose batch tree orthogonal matching pursuit (BTOMP) (Algorithm 2) that organizes the dictionary using a tree structure. BTOMP uses K-Means to group the dictionary into the $o$ sub-dictionaries $\{D_{g_1}, \cdots, D_{g_o}\}$, and associates the sub-dictionaries with the learned centers $C = [c_1, \cdots, c_o]$. The filter is selected in two steps: (1) select the center that best matches the current residual and (2) choose the filter within the sub-dictionary associated with this center. BTOMP reduces the cost of the filter selection to $O(oK + \frac{mh}{o})$ and the memory to $O(om)$. BTOMP uses a tree based approximate algorithm to select the filter, and we have found that it works well in practice. If $o = m$, BTOMP exactly recovers the batch orthogonal matching pursuit.

**Spatial Pyramid Max Pooling.** Spatial pyramid max pooling is a highly nonlinear operator that generates higher level representations from sparse codes of local patches which are spatially close. It aggregates these sparse codes using max pooling in a multi-level patch decomposition. At level 0, the decomposition consists of just a single spatial cell (whole patch). At level 1, the patch is subdivided into four quadrants, yielding four feature vectors, and so on. Let $U$ be the number of pyramid levels, $V_u$ the number of spatial cells in the $u$-th pyramid level, and $P$ be an image cell, then max pooling at the spatial cell $P$ can be represented as

$$F(P) = \left[ \max_{j \in P} |x_{j1}|, \cdots, \max_{j \in P} |x_{jh}| \right] \tag{5}$$

Concatenating max pooling features from different spatial cells, we have the patch-level feature: $F(P) = [F(P_1^1), \cdots, F(P_1^{V_1}), \cdots, F(P_U^{V_U})]$.

**Contrast Normalization.** The magnitude of sparse codes varies over a wide range due to local variations in illumination and foreground-background contrast, so effective local contrast normalization turns out to be essential for good recognition performance. We have compared two normalization schemes: $L_1$ normalization and $L_2$ normalization and found that the latter is consistently better than the former. For an image patch $P$, the $L_2$ normalization has the form

$$\overline{F}(P) = \frac{F(P)}{\sqrt{\|F(P)\|^2 + \epsilon}} \tag{6}$$

where $\epsilon$ is a small positive number. We have experimented with different $\epsilon$ values. We found that the best $\epsilon$ value in the first layer is around 0.1. Image intensity is always normalized to [0, 1] in our experiments. This is intuitive because a small threshold is able to make low contrast patches more separate from high contrast image patches, increasing the discrimination of features. In the deeper layers, recognition performance is robust to the $\epsilon$ value as long as it is small enough (for example $< 10^{-6}$).

### 2.3 Hierarchical Matching Pursuit

The matching pursuit encoder in the second layer is built on top of the outputs of the matching pursuit encoder in the first layer. Training is accomplished in a greedy, layer-wise way: once a lower

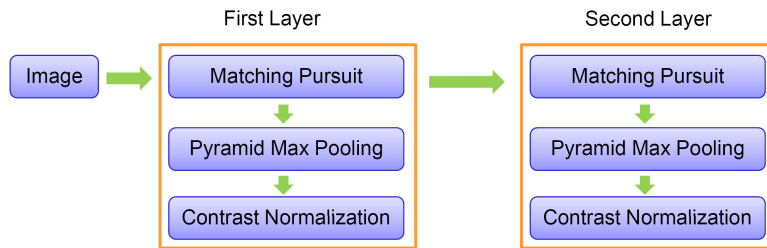

Figure 1: Hierarchical Matching Pursuit. In the first layer, sparse codes from small image patches are aggregated into patch-level features. In the second layer, sparse codes from patch-level features are aggregated across the whole image to produce image-level features. Batch tree orthogonal matching pursuit is used to compute sparse codes in each layer.

layer is trained, its dictionary is fixed, and its outputs are used as inputs to the next layer. We enforce that the patch size in the second layer is larger than that in the first layer, which makes sure that a higher level representation is extracted in the higher layer. More layers can be appended in a similar way to produce deep representations.

## 3 Experiments

We compare hierarchical matching pursuit with many state-of-the-art image classification algorithms on three publicly available datasets: Caltech101, MIT-Scene, and UIUC-Sports. Image intensity is normalized to [0, 1]. All images are transformed into grayscale and resized to be no larger than $300 \times 300$ pixels with preserved ratio.

We use two-layer hierarchical matching pursuit in all experiments. We have experimented with one-layer and three-layer HMP, but found that one-layer HMP is much worse than two-layer HMP while three-layer HMP doesn't improve recognition performance substantially. We learn the dictionary in the two layers by performing K-SVD on 1,000,000 sampled patches. In the first layer, we remove the zero frequency component from image patches by subtracting their means, and initialize K-SVD with the overcomplete discrete cosine transform (DCT) dictionary. Our pre-processing is simpler than other feature learning approaches that normalize image patches by dividing the standard deviation and then whitening the normalized image patches [6]. In the second layer, we initialize K-SVD with randomly sampled patch features. We set the number of the filters to be 3 times the filter size in the first layer and to be 1000 in the second layer. We use batch orthogonal matching pursuit to compute sparse codes. We set the sparsity level $K$ in the two layers to be 5 and 10, respectively.

We perform max pooling in a 3-level spatial pyramid, partitioned into $1 \times 1$, $2 \times 2$, and $4 \times 4$ sub-regions. In the first layer, we run the matching pursuit encoder on $16 \times 16$ image patches over dense grids with a step size of 4 pixels. In the second layer, we run the matching pursuit encoder on the whole image to produce the image-level features. For computational efficiency, we perform our spatial pyramid max pooling across the image with a step size of 4 pixels, rather than at each pixel. Given the high dimensionality of the learned features, we train linear SVM classifiers for image classification. Our experiments show that the linear SVM matches the performance of a nonlinear SVM with a histogram intersection kernel, which is consistent with the observations in [24, 5]. This allows our system to scale to large datasets. The regularization parameter in linear SVM is fixed to 10 in all the experiments. The filter size in the first layer is optimized by 5-fold cross validation on the training set.

We compare HMP to SIFT based single layer sparse coding because of its success in both computer vision and machine learning communities [24, 23, 5, 6]. We extract SIFT with $16 \times 16$ image patches over dense regular grids with spacing of 8 pixels. We use the publicly available dense SIFT code at http://www.cs.unc.edu/~lazebnik [14]. We perform sparse coding feature extraction using 1,000 visual words learned from 1,000,000 SIFT features, and compute image-level features by running spatial pyramid max pooling on $1 \times 1$, $2 \times 2$ and $4 \times 4$ sub-regions [24].

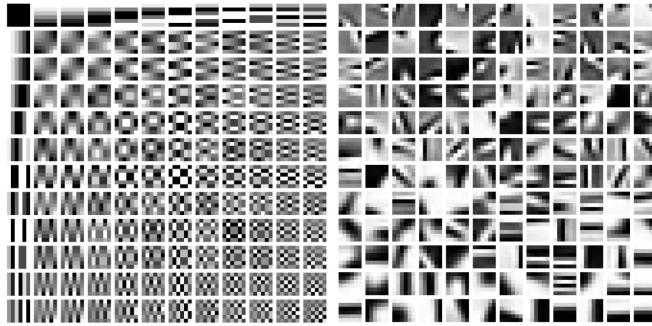

Figure 2: **Left:** The overcomplete DCT dictionary with 144 filters of size $6 \times 6$. **Right:** The dictionary with 144 filters of size $6 \times 6$ learned by K-SVD. It can be seen that the filters learned by K-SVD is much more diverse than those generated by the overcomplete DCT.

| Methods | 3×3 | 4×4 | 5×5 | 6×6 | 7×7 | 8×8 |
|---|---|---|---|---|---|---|
| DCT (orthogonal) | 69.9±0.6 | 70.8±0.3 | 71.5±1.0 | 72.1±0.7 | 73.2±0.4 | 73.1±0.7 |
| DCT (overcomplete) | 69.6±0.6 | 71.8±0.6 | 73.0±0.7 | 74.1±0.4 | 73.7±0.6 | 73.4±0.8 |
| K-SVD | 71.8±0.5 | 74.4±0.6 | 75.9±0.7 | **76.8±0.4** | 76.3±0.4 | 76.1±0.5 |

Table 1: Classification accuracy with different filter sizes.

## 3.1 Object Recognition

Caltech-101 contains 9,144 images from 101 object categories and one background category. Following the standard experimental setting, we train models on 30 images and test on no more than 50 images per category.

**Filter Size in the First Layer.** We show recognition accuracy as a function of the filter size in Table 1. The other parameters are fixed to the default values. We consider the orthogonal and overcomplete DCT dictionaries, and the overcomplete K-SVD dictionary. We have found that the orthogonal DCT achieves the highest accuracy when all the filters are chosen (without sparsity), and the overcomplete DCT and K-SVD have good accuracy at the sparsity level $T = 5$. We keep the overcomplete DCT dictionary and the K-SVD dictionary to have roughly similar sizes. From Table 1, we see that the orthogonal DCT dictionary works surprisingly well, and is very competitive with current state-of-the-art feature learning algorithms (see Table 3). The overcomplete K-SVD dictionary performs consistently better than the DCT dictionary. The best filter size of K-SVD is $6 \times 6$, which gives **76.8**% accuracy on this dataset, about 3% higher than the overcomplete DCT. We show the overcomplete DCT dictionary and the K-SVD dictionary in Fig. 2. As we see, the K-SVD dictionary not only includes the edge and dot filters, but also texture, multi-peaked, and high frequency filters, and is much more diverse than the overcomplete DCT dictionary.

**Spatial Pyramid Pooling.** Spatial pyramid max pooling introduces the different levels of spatial information, and always outperforms flat spatial max pooling ($4 \times 4$) by about 2% in our experiments.

**Contrast Normalization.** We evaluated HMP with and without contrast normalization. Our experiments show that contrast normalization improves recognition accuracy by about 3%, which suggests this is a very useful module for feature learning.

**Sparsity.** We show recognition accuracy as a function of the sparsity level $K$ in Fig. 3. The filter size is $6 \times 6$. When sparsity level in first or second layer varies, the other parameters are fixed to the default setting. We see that the accuracy is more robust to the zero-norm in the first layer while more sensitive in the second layer. The optimal $K$ in the two layers is around 5 and 10, respectively.

**Running Time.** The total cost of learning the dictionary using K-SVD is less than two hour. BOMP is about 10x faster in the second layer in our default setting, which dominates the running time of feature extraction. All experiments are run on a single 3.30GHz Intel Xeon CPU with a single thread. Efficient feature-sign search algorithm [15] is used to solve the sparse coding problem with an $L_1$ penalty. We compare the running cost of different algorithms for a typical $300 \times 300$ image in Table 2. HMP is much faster than single layer sparse coding and deconvolutional networks.

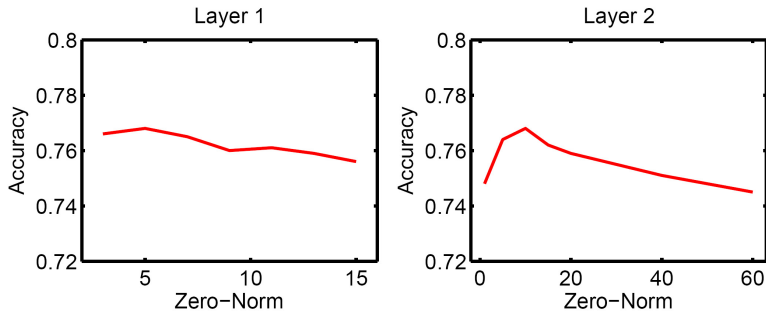

Figure 3: **Left:** Recognition accuracy as a function of zero-norm in the first layer. **Right:** Recognition accuracy as a function of zero-norm in the second layer.

| Methods | HMP(DCT) | HMP(K-SVD) | SIFT+SC | DN |
|---|---|---|---|---|
| Time (seconds) | 0.4 | 0.8 | 22.4 | 67.5 |

Table 2: Feature extraction time on a typical $300 \times 300$ image. HMP(DCT) means that the orthogonal DCT dictionary is used in the first layer. HMP(K-SVD) means that the learned dictionary is used. SIFT+SC denotes single layer sparse coding based on SIFT features.

**Large Dictionary.** We compared BTOMP and BOMP on a large dictionary with 10,000 filters in the second layer. We found that BTOMP is about 5 times faster than BOMP when the number of sub-groups is set to be 1000. BTOMP and BOMP have the almost same accuracy (**77.2**%), higher than the standard setting (1000 filters in the second layer).

**Comparisons with State-of-the-art Approaches.** We compare HMP with recent single feature based approaches in Table 3. In the first two columns, we see that HMP performs much better than other hierarchial feature learning approaches: invariant predictive sparse decomposition (IPSD) [12, 13], convolutional deep belief networks (CDBN) [16], and deconvolutional networks (DN) [26]. In the middle two columns, we show that HMP outperforms single layer sparse coding approaches on top of SIFT features: soft threshold coding (SIFT+T) [6], locality-constrained linear coding (LLC) [23] and Macrofeatures based sparse coding [5], and hierarchical sparse coding [25]. Notice that LLC is the best-performing approach in the first ImageNet Large-scale Visual Recognition Challenge [23]. In the right two columns, we compare HMP with naive Bayesian nearest neighbor (NBNN) [4] and three representative kernel methods: spatial pyramid matching (SPM) [14], metric learning for CORR kernel (ML+CORR) [10], and gradient kernel descriptors (KDES-G) [3, 2]. This group of approaches are based on SIFT features except for gradient kernel descriptors that extract patch-level features using weighted sum match kernels. Hierarchical matching pursuit is more than 10% better than SPM, a widely accepted baseline, and slightly better than NBNN and KDES-G in terms of accuracy. To our best knowledge, our feature learning system has the highest accuracy among single feature based approaches. Slightly higher accuracy (around 80%) has been reported with multiple kernel learning that combines many different types of image features [8].

| HMP | **76.8±0.4** | SIFT+T [6] | 67.7 | SPM [14] | 64.4 |
|---|---|---|---|---|---|
| IPSD [12] | 65.5 | HSC [25] | 74.0 | ML+CORR [10] | 69.6 |
| CDBN [16] | 65.4 | LLC [23] | 73.4±0.5 | NBNN [4] | 73.0 |
| DN [26] | 66.9±1.1 | Macrofeatures [5] | 75.7±1.1 | KDES-G [2] | 75.2±0.4 |

Table 3: Comparisons on Caltech-101. Hierarchical matching pursuit is compared to recently published object recognition algorithms.

## 3.2 Scene Recognition

We evaluate hierarchical matching pursuit for scene recognition on the MIT-Scene dataset [20]. This dataset contain 15620 images from 67 indoor scene categories. All images have a minimum resolution of 200 pixels in the smallest axis. This recognition task is very challenging since the large in-class variability and small between-class variability in this dataset (see Figure 4). Following the

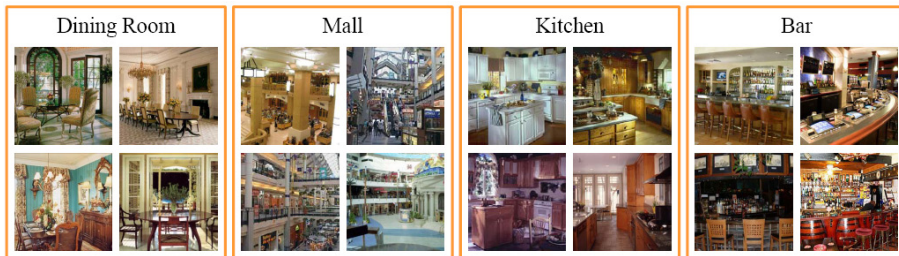

Figure 4: Sampled scene categories from the 67 indoor scene.

| Methods | HMP | OB [18] | GIST [20] | ROI+GIST [20] | SIFT+SC |
|---|---|---|---|---|---|
| Accuracy | **41.8** | 37.6 | 22.0 | 26.0 | 36.9 |

Table 4: Comparisons on the MIT-Scene dataset. OB denotes the object bank approach proposed in [18]. ROI denotes region of interest. SIFT+SC has similar performance with SIFT+OMP.

standard experimental setting [20], we train models on 80 images and test on 20 images per category. We report the accuracy of HMP over the training/test split provided on the authors's website in Table 4. HMP has an accuracy of **41.8**% with the filter size $4 \times 4$, more than 15 percent higher than GIST features based approach, and about 5 percent higher than SIFT based sparse coding and object bank. Object bank is a recently proposed high-level feature, which trains 200 object detectors using the object bounding boxes from the LabelMe and ImageNet dataset, and runs them across an image at different scales to produce image features. To the best of our knowledge, this accuracy is beyond all previously published results on this data.

### 3.3 Event Recognition

We evaluate hierarchical matching pursuit for static event recognition on the UIUC-Sports dataset [18]. This dataset consists of 8 sport event categories: rowing, badminton, polo, bocce, snowboarding, croquet, sailing and rock climbing with 137 to 250 images in each. Following the common experimental setting [18], we train models on 70 images and test on 60 images per category. We report the averaged accuracy of HMP over 10 random training/test splits in Table 5. The optimal filter size is $4 \times 4$. As we see, HMP significantly outperforms SIFT based generative graphical model, SIFT based single layer sparse coding, and the recent object bank approach significantly. The accuracy obtained by HMP is the best published result on this dataset to date.

| Methods | HMP | OB [18] | SIFT+GGM [17] | SIFT+SC |
|---|---|---|---|---|
| Accuracy | **85.7±1.3** | 76.3 | 73.4 | 82.7±1.5 |

Table 5: Comparisons on the UIUC-Sports dataset. GGM denotes the generative graphical model proposed in [17].

## 4 Conclusion

We have proposed *hierarchical matching pursuit*, to learn meaningful multi-level representations from images layer by layer. Hierarchical matching pursuit uses the matching pursuit encoder to build a feature hierarchy that consists of three modules: batch tree orthogonal matching pursuit, spatial pyramid matching, and contrast normalization. Our system is scalable, and can efficiently handle full-size images. In addition, we have proposed batch tree orthogonal matching pursuit to speed up feature extraction at runtime. We have performed extensive comparisons on three types of image classification tasks: object recognition, scene recognition, and event recognition. Our experiments have confirmed that hierarchical matching pursuit outperforms both SIFT based single layer sparse coding and other hierarchical feature learning approaches: convolutional deep belief networks, convolutional neural networks and deconvolutional networks.

**Acknowledgements.** This work was funded in part by an Intel grant and by ONR MURI grants N00014-07-1-0749 and N00014-09-1-1052.

# References

[1] M. Aharon, M. Elad, and A. Bruckstein. K-SVD: An Algorithm for Designing Overcomplete Dictionaries for Sparse Representation. *IEEE Transactions on Signal Processing*, 54(11):4311–4322, 2006.

[2] L. Bo, X. Ren, and D. Fox. Kernel Descriptors for Visual Recognition. In *NIPS*, 2010.

[3] L. Bo and C. Sminchisescu. Efficient Match Kernel between Sets of Features for Visual Recognition. In *NIPS*, 2009.

[4] O. Boiman, E. Shechtman, and M. Irani. In Defense of Nearest-Neighbor based Image Classification. In *CVPR*, 2008.

[5] Y. Boureau, F. Bach, Y. LeCun, and J. Ponce. Learning Mid-level Features for Recognition. In *CVPR*, 2010.

[6] A. Coates and A. Ng. The Importance of Encoding versus Training with Sparse Coding and Vector Quantization. In *ICML*, 2011.

[7] G. Davis, S. Mallat, and M. Avellaneda. Adaptive Greedy Approximations. *Constructive Approximation*, 13(1):57–98, 1997.

[8] P. Gehler and S. Nowozin. On Feature Combination for Multiclass Object Classification. In *ICCV*, 2009.

[9] G. Hinton, S. Osindero, and Y. Teh. A Fast Learning Algorithm for Deep Belief Nets. *Neural Computation*, 18(7):1527–1554, 2006.

[10] P. Jain, B. Kulis, and K. Grauman. Fast Image Search for Learned Metrics. In *CVPR*, 2008.

[11] K. Jarrett, K. Kavukcuoglu, M. Ranzato, and Y. LeCun. What is the Best Multi-Stage Architecture for Object Recognition? In *ICCV*, 2009.

[12] K. Kavukcuoglu, M. Ranzato, R. Fergus, and Y. LeCun. Learning Invariant Features through Topographic Filter Maps. In *CVPR*, 2009.

[13] K. Kavukcuoglu, P. Sermanet, Y. Boureau, K. Gregor, M. Mathieu, and Y. LeCun. Learning Convolutional Feature Hierarchies for Visual Recognition. In *NIPS*. 2010.

[14] S. Lazebnik, C. Schmid, and J. Ponce. Beyond Bags of Features: Spatial Pyramid Matching for Recognizing Natural Scene Categories. In *CVPR*, 2006.

[15] H. Lee, A. Battle, R. Raina, and A. Ng. Efficient Sparse Coding Algorithms. In *NIPS*. 2007.

[16] H. Lee, R. Grosse, R. Ranganath, and A. Ng. Convolutional Deep Belief Networks for Scalable Unsupervised Learning of Hierarchical Representations. In *ICML*, 2009.

[17] L. Li and L. Fei-Fei. What, Where and Who? Classifying Event by Scene and Object Recognition. In *ICCV*, 2007.

[18] L. Li, H. Su, E. Xing, and L. Fei-Fei. Object Bank: A High-Level Image Representation for Scene Classification and Semantic Feature Sparsification. In *NIPS*, 2010.

[19] Y. Pati, R. Rezaiifar, and P. Krishnaprasad. Orthogonal Matching Pursuit: Recursive Function Approximation with Applications to Wavelet Decomposition. In *The Twenty-Seventh Asilomar Conference on Signals, Systems and Computers*, pages 40–44, 1993.

[20] A. Quattoni and A. Torralba. Recognizing Indoor Scenes. In *CVPR*, 2009.

[21] R. Rubinstein, A. Bruckstein, and M Elad. Dictionaries for Sparse Representation Modeling. *Proceedings of the IEEE*, 98(6):4311–4322, 2010.

[22] R. Rubinstein, M. Zibulevsky, and M. Elad. Efficient Implementation of the K-SVD Algorithm using Batch Orthogonal Matching Pursuit. Technical report, 2008.

[23] J. Wang, J. Yang, K. Yu, F. Lv, T. Huang, and Y. Guo. Locality-constrained Linear Coding for Image Classification. In *CVPR*, 2010.

[24] J. Yang, K. Yu, Y. Gong, and T. Huang. Linear Spatial Pyramid Matching using Sparse Coding for Image Classification. In *CVPR*, 2009.

[25] K. Yu, Y. Lin, and J. Lafferty. Learning Image Representations from the Pixel Level via Hierarchical Sparse Coding. In *CVPR*, 2011.

[26] M. Zeiler, D. Krishnan, G. Taylor, and R. Fergus. Deconvolutional Networks. In *CVPR*, 2010.

